# Linear readout from a neural population with partial correlation data

**Adrien Wohrer**[(1)]**, Ranulfo Romo**[(2)]**, Christian Machens**[(1)]

[(1)] Group for Neural Theory
Laboratoire de Neurosciences Cognitives
École Normale Suprieure
75005 Paris, France
{adrien.wohrer,christian.machens}@ens.fr

[(2)] Instituto de Fisiología Celular
Universidad Nacional Autónoma de México
Mexico City, Mexico
rromo@ifc.unam.mx

## Abstract

How much information does a neural population convey about a stimulus? Answers to this question are known to strongly depend on the correlation of response variability in neural populations. These noise correlations, however, are essentially immeasurable as the number of parameters in a noise correlation matrix grows quadratically with population size. Here, we suggest to bypass this problem by imposing a parametric model on a noise correlation matrix. Our basic assumption is that noise correlations arise due to common inputs between neurons. On average, noise correlations will therefore reflect signal correlations, which can be measured in neural populations. We suggest an explicit parametric dependency between signal and noise correlations. We show how this dependency can be used to "fill the gaps" in noise correlations matrices using an iterative application of the Wishart distribution over positive definitive matrices. We apply our method to data from the primary somatosensory cortex of monkeys performing a two-alternative-forced choice task. We compare the discrimination thresholds read out from the population of recorded neurons with the discrimination threshold of the monkey and show that our method predicts different results than simpler, average schemes of noise correlations.

## 1 Introduction

In the field of population coding, a recurring question is the impact on coding efficiency of so-called *noise correlations*, *i.e.*, trial-to-trial covariation of different neurons' activities due to shared connectivity. Noise correlations have been proposed to be either detrimental or beneficial to the quantity of information conveyed by a population [1, 2, 3]. Also, some proposed neural coding schemes, such as those based on synchronous spike waves, fundamentally rely on second- and higher- order correlations in the population spikes [4].

The problem of noise correlations is made particularly difficult by its high dimensionality along two distinct physical magnitudes: time, and number of neurons. Ideally, one should describe the probabilistic structure of any set of spike trains, at any times, for any ensemble of neurons in the population; which is clearly impossible experimentally. As a result, when recording from a

population of neurons with a finite number of trials, one only has access to very *partial* correlation data. First, studies based on experimental data are most often limited to second order (pairwise) correlations. Second, the temporal correlation structure is generally simplified (*e.g.*, by assuming stationarity) or forgotten altogether (by studying only correlation in overall spike counts). Third and most importantly, even with modern multi-electrode arrays, one is limited in the number of neurons which can be recorded simultaneously during an experiment. Thus, when data are pooled over experiments involving different neurons, most pairwise noise correlation indices remain unknown. In consequence, there is always a strong need to "fill the gaps" in the partial correlation data extracted experimentally from a population.

In contrast to noise-correlation data, the *first-order* probabilistic data are easily extracted from a population: They simply consist in the *trial-averaged firing rates* of the neurons, generally referred to as their "signal". In particular, one can easily measure so-called *signal correlations* which measure how different neurons' trial-averaged firing rates covary with changes in the stimulus.

In this paper, we propose a method to "fill the gaps" in noise correlation data, based on signal correlation data. This approach can be summarized by the notion that "similar tuning reveals shared inputs". Indeed, noise correlations reveal a proximity of connection between neurons (through shared inputs and/or reciprocal connections) which, in turn, will generally result in some covariation of the neurons' first-order response to stimuli. When browsing through neural pairs in the population, one should thus expect to find a statistical link between their signal- and noise- correlations; and this has indeed been reported several times [5, 6]. If this statistical structure is well described, it can serve as basis to *randomly* generate noise correlation structures, compatible with the measured signal correlation. Furthermore, to assess the impact of this randomness, one can perform repeated picks of potential noise correlation structures, each time observing the resulting impact on the coding capacity in the population. Then, this method will provide reliable estimates (average + error bar) of the impact of noise correlations on population coding, given partial noise correlation data.

We present this general approach in a simplified setting in Section 2. The input stimulus is a single parameter which can take a finite number of values. The population's response is summarized by a single number for each neuron (its mean firing rate during the trial), so that in turn a correlation structure is simply given by a symmetric, positive, $N$x$N$ matrix. In Section 3, we detail the method used to generate random noise correlation matrices compatible with the population's signal correlation, which we believe to be novel. In Section 4, we apply this procedure to assess the amount of information about the stimulus in the somatosensory cortex of macaques responding to tactile stimulation.

## 2   Model of the neural population

**Population activity R.**   We consider a population of $N$ neurons tested over a discrete set of possible stimuli $f \in \{f_1, \ldots, f_K\}$, lasting for a period of time $T$. The spike train of neuron $i$ can be described by a series of Dirac pulses $S_i(t) = \sum_{k=1}^{n_i} \delta(t - t_k^{(i)})$. Due to trial-to-trial variability, the number of emitted spikes $n_i$ and the spike times $t_k^{(i)}$ are random variables, whose distribution depends (amongst other things) on the value of stimulus $f$.

At each trial, information about $f$ can be extracted from the spike trains $S_i(t)$ using several possible readout mechanisms. In this article, we limit ourselves to the simplest type of readout: The population activity is summarized by the $N$-dimensional vector $\mathbf{R} = \{R_i\}_{i=1\ldots N}$, where $R_i = n_i/T$ is the mean firing rate of neuron $i$ on this trial. A more plausible readout, based on sliding-window estimates of the instantaneous firing rate, has been presented elsewhere [7].

**First-moment measurements.**   Given a particular stimulus $f$, we note $\lambda_i(t, f)$ the probability of observing a spike from neuron $i$ at time $t$ regardless of other neurons' spikes (*i.e.*, the *first moment density*, in the nomenclature of point processes): $E(S_i(t) \mid f) = \lambda_i(t, f)$. Experimentally, $\lambda_i(t, f)$ is measured fairly easily, as the *trial-averaged firing rate* of neuron $i$ in stimulus condition $f$.

Since $R_i = 1/T \sum_{t=0}^{T} S_i(t)$, its expectancy is given by

$$E(R_i \,|\, f) = 1/T \sum_{t=0}^{T} \lambda_i(t,f) \stackrel{\Delta}{=} \overline{\lambda_i}(f). \tag{1}$$

This function of $f$ is generally called the *tuning curve* of neuron $i$.

The trial-averaged firing rates $\lambda_i(t,f)$ can also be used to define the *signal correlation* matrix $\boldsymbol{\sigma} = \{\sigma_{ij}\}_{i,j=1...N}$, as:

$$\sigma_{ij} = \frac{\sum_{f,t} \lambda_i(t,f)\lambda_j(t,f) - KT\widehat{\lambda_i}\widehat{\lambda_j}}{\sqrt{\left(\sum_{f,t} \lambda_i(t,f)^2 - KT\widehat{\lambda_i}^2\right)\left(\sum_{f,t} \lambda_j(t,f)^2 - KT\widehat{\lambda_j}^2\right)}},$$

where $\widehat{\lambda_i} = 1/(KT)\sum_{f,t} \lambda_i(t,f)$ is the overall average firing rate of neuron $i$ across trials and stimuli. The Pearson correlation $\sigma_{ij}$ measures how much the first-order responses of neurons $i$ and $j$ "look alike", both in their temporal course and across stimuli. Being a correlation matrix, $\boldsymbol{\sigma}$ is positive definite, with 1s on its diagonal, and off-diagonal elements between $-1$ and 1. As opposed to most studies which define signal correlation only based on tuning curves, it is important for our purpose to also include the *time course* of response in the measure of signal similarity. Indeed, similar temporal courses are more likely to reveal shared input, and thus possible noise correlation.

**A model for noise correlations.** While first-moment ("signal") statistics can be measured experimentally with good precision, second-moment statistics (noise correlations) can never be totally measured in a large population. For this reason a parametric model must be introduced, that will allow us to infer the correlation parameters that could not be measured.

We introduce a simple model in which the noise correlation matrix $\boldsymbol{\rho}$ is independent of stimulus $f$: For a given stimulus $f$, the population activity $\mathbf{R}$ is supposed to follow the multivariate Gaussian $\mathcal{N}(\boldsymbol{\mu}(f), \mathbf{Q}(f))$, with

$$\mu_i(f) = \overline{\lambda_i}(f), \tag{2}$$

$$Q_{ij}(f) = \rho_{ij}\sqrt{\overline{\lambda_i}(f)\overline{\lambda_j}(f)}. \tag{3}$$

Let us make a few remarks about this model. The first line is imposed by eq. (1). The second line implies that $\mathrm{var}(R_i \,|\, f) = Q_{ii}(f) = \overline{\lambda_i}(f)$, meaning that all neurons in this model are supposed to have a Fano factor of one. This model is the simplest possible for our purpose, as its only free parameter is the chosen noise correlation matrix $\boldsymbol{\rho}$, and it has often been used in the literature [8]. Naturally, the assumption of Gaussianity is a simplifying approximation, as the values for $\mathbf{R}$ really come from a discretized spike count.

## 3 Inferring the full noise correlation structure

### 3.1 Statistical link between *signal* and *noise* correlation

We propose that, across all pairs $(i,j)$ of distinct cells in the population, the noise correlation index is linked to the signal correlation index by the following statistical relationship:

$$\rho_{ij} \sim \mathcal{N}\big(F(\sigma_{ij}), c^2\big), \tag{4}$$

where function $F(\sigma_{ij})$ provides the expected value for $\rho_{ij}$ if $\sigma_{ij}$ is known, and $c$ measures the statistical variations of $\rho_{ij}$ across pairs of cells sharing the same signal correlation $\sigma_{ij}$. By extension, we note $F(\boldsymbol{\sigma})$ the matrix with 1s on its diagonal, and non-diagonal elements $F(\sigma_{ij})$.

The choice of $F$ and $c$ is dictated by the experimental data under study. In our case, these are neural recordings in the primary somatosensory cortex (S1) of monkeys responding to a frequency discrimination task (see Section 4). For all pairs $(i,j)$ of simultaneously recorded neurons (total of several hundred pairs), we computed the two correlation coefficients $(\sigma_{ij}, \rho_{ij})$. This allowed us to compute an experimental estimate for the distribution of $\rho_{ij}$ given $\sigma_{ij}$ (Figure 1). We find that

$$F(x) = b + a\,\exp\big(\alpha(x-1)\big) \tag{5}$$

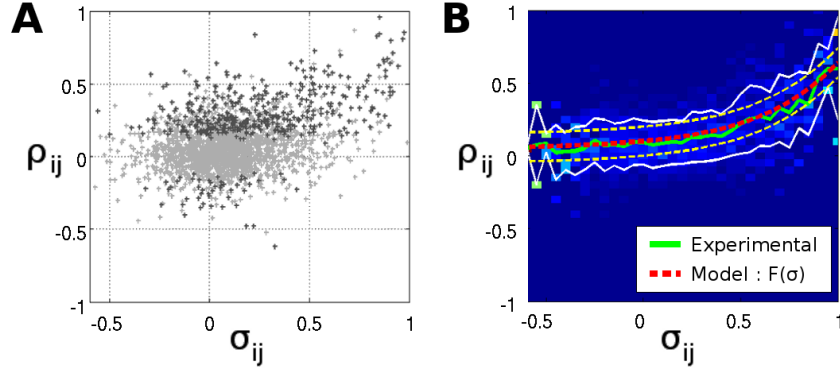

Figure 1: *Statistical link between signal and noise correlations.* A: Experimental distribution of $(\sigma_{ij}, \rho_{ij})$ across simultaneously recorded neural pairs in population data from cortical area S1 (*dark gray*: noise correlation coefficients significantly different from 0). B: Same data transformed into a conditional distribution for $\rho_{ij}$ given $\sigma_{ij}$. Plain ligns: experimental mean (*green*) and error bars (*white*). Dotted ligns: model mean $F(\sigma_{ij})$ (*red*) and standard deviation $c$ (*yellow*).

provides a good fit, with $a \simeq 0.6$, $\alpha \simeq 2.5$ and $b \simeq 0.05$. For the standard deviation in eq. (4), we choose $c = 0.1$. This value is slightly reduced compared to experimental data (Figure 1, white vs. yellow confidence intervals), because part of the variability of $\rho_{ij}$ observed experimentally is due to finite-sample errors in its measurement. We also note that the value found here for $a$ is higher than values generally reported for noise correlations in the literature [2], possibly due to experimental limitations ; however, this has no influence on the method proposed here, only on its quantitative results.

Once that function $F$ is fitted on the subset of simultaneously recorded neural pairs, we can use the statistical relation (4)-(5) to randomly generate noise correlation matrices $\boldsymbol{\rho}$ for the full neural population, on the basis of its signal correlation matrix $\boldsymbol{\sigma}$. However, such a random generation is not trivial, as one must insure at the same time that individual coefficients $\rho_{ij}$ follow relation (4), and that $\boldsymbol{\rho}$ remains a (positive definite) correlation matrix.

As a first step towards this generation, note that the "average" noise correlation matrix predicted by the model, that is $F(\boldsymbol{\sigma})$, is itself a correlation matrix. First, by construction, it has 1s on the diagonal and all its elements belong to $[-1, 1]$. Second, $F(\boldsymbol{\sigma})$ can be written as a Taylor expansion on element-wise powers of $\boldsymbol{\sigma}$ (plus diagonal term $(1-a-b)\mathrm{Id}$), with only positive coefficients (due to the exponential in eq. (5)). Since the element-wise (or *Hadamard*) product of two symmetric semi-definite positive matrices is itself semi-positive definite ("Schur's product theorem" [9]), all matrices in the expansion are semi-definite positive, and so is $F(\boldsymbol{\sigma})$. This property is fundamental to apply the method of random matrix generation that we propose now.

## 3.2 Generating random correlation matrices

**Wishart and *anti-Wishart* distributions.** The Wishart distribution is probably the most straightforward way of generating a random symmetric, positive definite matrix with an imposed expectancy matrix. Let $\Sigma$ be an $N \mathrm{x} N$ symmetric definite positive matrix, $k$ an integer giving the number of degrees of freedom, and introduce the sample covariance matrix of $k$ i.i.d Gaussian samples $\mathbf{X}_i$ drawn according to $\mathcal{N}(\mathbf{0}, \Sigma)$: $\boldsymbol{\Omega} = 1/k \sum_{i=1}^{k} \mathbf{X}_i \mathbf{X}_i^T$. When $k \geq N$, the matrix $\boldsymbol{\Omega}$ has almost surely full-rank. In that case, its pdf has a relatively simple expression, and the distribution for $\boldsymbol{\Omega}$ is called the Wishart distribution [10]. When $k < N$, the matrix $\boldsymbol{\Omega}$ is almost surely of rank $k$, so it is not invertible anymore. In that case, its pdf has a much more intricate expression. This distribution has sometimes been referred to as *anti-Wishart* distribution [11].

In both cases, the resulting distribution for random matrix $\mathbf{\Omega}$, which we note $\mathcal{W}(\Sigma, k)$, can be proven to have the following characteristic function [11]:

$$\phi(T) = \mathrm{E}(\mathrm{e}^{-i\mathrm{Tr}(\mathbf{\Omega}T)}) = \det\left(\mathrm{Id} + \frac{2i}{k}\Sigma T\right)^{k/2}$$

(where $T$ is a real symmetric matrix). This result can be used to find the two first moments of $\mathbf{\Omega}$:

$$\mathrm{E}(\Omega_{ij}) = \Sigma_{ij} \tag{6}$$

$$\mathrm{cov}(\Omega_{ij}, \Omega_{kl}) = \frac{1}{k}(\Sigma_{ik}\Sigma_{jl} + \Sigma_{il}\Sigma_{jk}), \tag{7}$$

with a variance naturally scaling as $1/k$.

Then, a second step consists in renormalizing $\mathbf{\Omega}$ by its diagonal elements, to produce a correlation matrix $\boldsymbol{\rho}$. The resulting distribution for $\boldsymbol{\rho}$, which we note $\overline{\mathcal{W}}(\Sigma, k)$, has been studied by Fisher and others [12, 10], and is quite intricate to describe analytically. If one takes the generating matrix $\Sigma = F(\boldsymbol{\sigma})$ to be itself a correlation matrix, then $\mathrm{E}(\boldsymbol{\rho}) \simeq F(\boldsymbol{\sigma})$ still holds approximately, albeit with a small bias, and the variance of $\boldsymbol{\rho}$ still scales with $1/k$.

Distribution $\overline{\mathcal{W}}(F(\boldsymbol{\sigma}), k)$ could be a good candidate to generate a random correlation matrix $\boldsymbol{\rho}$ that would approximately verify $E(\boldsymbol{\rho}) = F(\boldsymbol{\sigma})$. Unfortunately, this method presents a problem in our case. To fit the statistical relation eq. (4), we need the variance of an element $\Omega_{ij}$ to be on the order of $c^2 \simeq 0.01$. But this implies (through eq. 7) that $k$ must be small (typically, around 20), so that noise correlation matrices $\boldsymbol{\rho}$ generated in this way necessarily have a very low rank (anti-Wishart distribution, Figure 2, blue traces). This creates an artificial feature of the noise correlation structure which is not at all desirable.

**Iterated Wishart.** We propose here an alternative method for generating random correlation matrices, based on *iterative* applications of the Wishart distribution. This method allows to create random correlation matrices with a higher variance than a Wishart distribution, while retaining a much wider eigenvalue spectrum than the more simple anti-Wishart distribution.

The distribution has two positive integer parameters $k$ and $m$ (plus generative matrix $F(\boldsymbol{\sigma})$). It is based on the following recursive procedure:

1. Start from deterministic matrix $\boldsymbol{\rho}_0 = F(\boldsymbol{\sigma})$.
2. For $n = 1 \ldots m$, pick $\boldsymbol{\rho}_n$ following the Wishart-correlation distribution $\overline{\mathcal{W}}(\boldsymbol{\rho}_{n-1}, k)$.
3. Take $\boldsymbol{\rho} = \boldsymbol{\rho}_m$ as output random matrix.

Since $\mathrm{E}(\boldsymbol{\rho}_n) \simeq \mathrm{E}(\boldsymbol{\rho}_{n-1})$, one expects approximately $\mathrm{E}(\boldsymbol{\rho}) \simeq F(\boldsymbol{\sigma})$. Furthermore, by taking a large $k$, one can produce full-rank matrices, circumventing the "low-rank problem" of the anti-Wishart distribution. Because $k$ is large, the variance added at each step is small (proportional to $1/k$), which is compensated by iterating the procedure a large number $m$ of times.

Simulations allowed us to study the resulting distribution for $\boldsymbol{\rho}$ (Figure 2, red traces) and compare it to the more standard "anti-Wishart-based" distribution for $\boldsymbol{\rho}$ (Figure 2, blue traces). We used the signal correlation data $\boldsymbol{\sigma}$ observed in a 100-neuron recorded sample from area S1, and the average noise correlation $F(\boldsymbol{\sigma})$ given by our experimental fit of $F$ in that same area (Figure 1). As a simple investigation into the expectancy and variance of these distributions, we computed the empirical distribution for $\rho_{ij}$ conditionned on $\sigma_{ij}$, for both distributions (Panel $A$). On this aspect the two distributions lead to very similar results, with a mean sticking closely to $F(\sigma_{ij})$, except for low values of $\sigma_{ij}$ where the slight bias, previously mentionned, is observed in both cases. In contrast, the two distributions lead to very different results in term of their spectra (Panel $B$). The iterative Wishart, used with a large value of $k$, preserves a non-null spectrum across all its dimensions. It should be noted, though, that the spectrum is markedly more concentrated on the first eigenvalues than the spectrum of $F(\boldsymbol{\sigma})$ (dotted line). However, this tendency towards dimensional reduction is much milder than in the anti-Wishart case !

As long as $m$ is sensibly smaller than $k$, the variances added at each step (of order $1/k$) simply sum up, so that $m/k$ is the main factor defining the variance of the distribution. For example, in

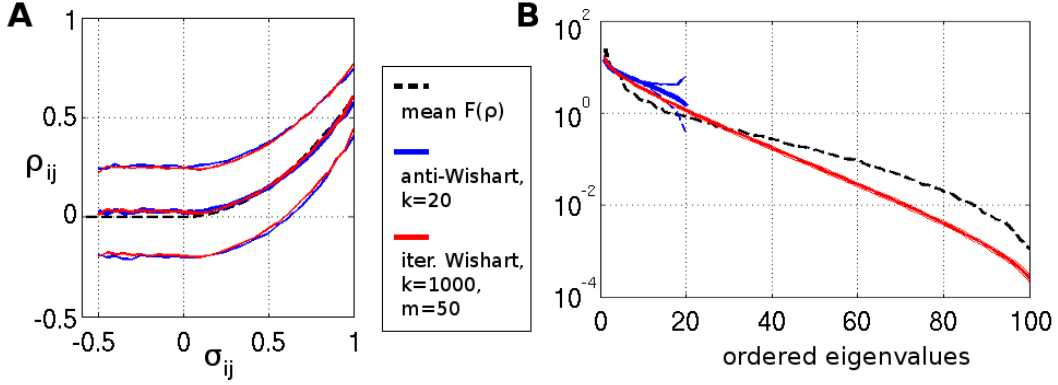

Figure 2: *Random generation of noise correlation matrices.* $N = 100$ neurons from our recorded sample (area S1). *A*: Empirical distribution of noise correlation $\rho_{ij}$ conditioned on signal correlation $\sigma_{ij}$ (mean $\pm$ std). *B*: Empirical distribution of eigenvalue spectrum (mean $\pm$ std in log domain).

Figure 2, $k/m$ equals 20, precisely the number of degrees of freedom in the equivalent anti-Wishart distribution. Also, the eigenvalue spectrum of $\boldsymbol{\rho}$ appears to follow a quasi-perfect exponential decay (even on a trial-by-trial basis), a result for which we have yet no explanation. The theoretical study of the "iterated Wishart" distribution, especially when $k$ and $m$ tend to infinity in a fixed ratio, might yield an interesting new type of distribution for positive symmetric matrices.

## 4   Linear encoding of tactile frequency in somatosensory cortex

To illustrate the interest of random noise correlation matrix generation, we come back to our experimental data. They consist of neural recordings in the somatosensory cortex of macaques during a two-frequency discrimination task. Two tactile vibrations are successively applied on the fingertips of a monkey. The monkey must then decide which vibration had the higher frequency (the detailed experimental protocol has been described elsewhere). Here, we analyze neural responses to the first presented frequency, in primary somatosensory cortex (S1). Most neurons there have a positive tuning ($\overline{\lambda_i}(f)$ grows with $f$) and positive noise correlations ; however, negative tunings (resulting in the appearance of negative signal correlations) and significant negative noise-correlations can also be found (Figure 1-*A*).

In the notations of Section 2, stimulus $f$ is the vibration frequency, which can take $K = 5$ possible values (14, 18, 22, 26 and 30 Hz). The neural activities $R_i$ consist of each neuron's mean firing rate over the duration of the stimulation, with $T = 250$ ms. Our goal is to estimate the amount of information about stimulus $f$ which can be extracted from a *linear readout* of neural activities, depending on the number of neurons $N$ in the population. This implies to estimate the impact of noise correlations. We thus generate a random noise correlation structure $\boldsymbol{\rho}$ following the above procedure, and assume the resulting distribution for neural activity $\mathbf{R}$ to follow eq. (2)-(3). This being given, one can estimate the sensitivity $\Delta f$ of a linear readout of $f$ from $\mathbf{R}$, as we now present.

### 4.1   Linear stimulus discriminability in a neural population

**Linear readout from the population.**   To predict the value of $f$ given $\mathbf{R}$, we resort to a simple one-dimensional linear readout, based on a prediction variable $\hat{f} = \sum_{i=1}^{N} a_i R_i$. The set of neural weights $\mathbf{a} = \{a_i\}_{i=1...N}$ must be chosen in order to maximize the readout performance. We find it through 1-dimensional Linear Discriminant Analysis (LDA), as the direction which maximizes $(\mathbf{a}^T \mathbf{M} \mathbf{a})/(\mathbf{a}^T \overline{\mathbf{Q}} \mathbf{a})$, where $\mathbf{M}$ is the inter-class covariance matrix of class centroids $\{\boldsymbol{\mu}(f)\}_{f=f_1...f_K}$, and $\overline{\mathbf{Q}} = 1/K \sum_f \mathbf{Q}(f)$ is the average intra-class covariance matrix. Then, the norm of $\mathbf{a}$ is chosen so as for variable $\hat{f}$ to be the best possible predictor of stimulus value $f$, in terms of mean square error.

**Readout discriminability.** The previous procedure produces a prediction variable $\hat{f}$ which is normally distributed, with $\mathrm{E}(\hat{f}\,|\,f) = \mathbf{a}^T\boldsymbol{\mu}(f)$ and $\mathrm{var}(\hat{f}\,|\,f) = \mathbf{a}^T\mathbf{Q}(f)\mathbf{a}$. As a result, one can compute analytically the *neurometric curve* giving the probability that two successive stimuli be correctly compared by the prediction variable:

$$G(\Delta) = P(\hat{f}_2 > \hat{f}_1 | f_2 - f_1 = \Delta). \tag{8}$$

Finally, a sigmoid can be fit to this curve and provide a single *neurometric index* $\Delta f$, as half its $25\% - 75\%$ interval. $\Delta f$ measures what we call the *linear discriminability* of stimulus $f$ in this neural population. It provides an estimate of the amount of information about the stimulus linearly present in the population activity $\mathbf{R}$.

## 4.2 Discriminability curves

**Discriminability versus population size.** The previous paragraphs have described a means to estimate the linear discriminability $\Delta f$ of a given neural population, with a given noise correlation structure. We apply this method to estimate $\Delta f(N)$ in growing populations of size $N = 1, 2, \ldots$, up to the full recorded neural sample (approx. 100 neurons in S1, Brodmann area 1). For each $N$, $\Delta f(N)$ is computed to approximate the linear discriminability of the *best* $N$-tuple population available from our recorded sample. As it is not tractable to test all possible $N$-tuples, we resort to the following recursive scheme: Search for neuron $i_1$ with best discriminability, then search for neuron $i_2$ with the best discriminability for 2-tuple $\{i_1, i_2\}$, etc. We term the resulting curve $\Delta f(N)$ the *discriminability curve* for the population. Note that this curve is not necessarily decreasing, as the last neurons to be included in the population can actually deteriorate the overall readout, by their influence on the LDA axis $\mathbf{a}$.

Each draw of a sample noise correlation structure gives rise to a different discriminability curve. To better assess the possible impact of noise correlations, we performed 20 random draws of possible noise correlation structures, each time computing the discriminability curve. This produces an *average* discrimination curve flanked by a confidence interval modelling our ignorance of the exact full correlation structure in the population (Figure 3, red lines). The confidence interval is found to be rather small. This means that, if our statistical model for the link between signal and noise correlation (4)-(5) is correct, it is possible to assess with good precision the content of information present in a neural population, even with very partial knowledge of its correlation structure.

Since the resulting confidence interval on $\Delta f(N)$ is small, one could assume that the impact of noise correlations is only driven by the "statistical average" matrix $F(\boldsymbol{\sigma})$. In this particular application, however, this is not the case. When the noise correlation matrix $\boldsymbol{\rho}$ is (deterministically) set equal to $F(\boldsymbol{\sigma})$, the resulting linear discriminability is underestimated (blue curve in Figure 3). Indeed, the statistical fluctuations in $\rho_{ij}$ around $F(\sigma_{ij})$, of magnitude $c \simeq 0.1$, induce an overcorrelation of certain neural pairs, and a decorrelation of other pairs (including a significant minority of negative correlation indices – as observed in our data, Figure 1). The net effect of the decorrelated pairs is stronger and improves the overall discriminability in the population as compared to the "statistical average".

In our particular case, the predicted discriminability curve is actually closer to what it would be in a totally decorrelated population ($\boldsymbol{\rho} = 0$, green curve). This result is not generic (it depends on the parameter values in this particular example), but it illustrates how noise correlations are not necessarily detrimental to coding efficiency [2], in neural populations with balanced tuning and/or balanced noise correlations (as is the case here, for a minority of cells).

**Comparison with monkey behavior.** The measure of discriminability through $G(\Delta)$ (eq. 8) mimics the two-stimulus comparison which is actually performed by the monkey. And indeed, one can build in the same fashion a *psychometric curve* for the monkey, describing its behavioral accuracy in comparing correctly $f_1$ and $f_2$ across trials, depending on $\Delta = f_2 - f_1$. The resulting *psychometric index* $\Delta f_{\text{monkey}}$ can then directly be compared with $\Delta f$, to assess the behavioral relevance of the proposed linear readout (Figure 3, black dotted line). In our model, the neurometric discriminability curve crosses the monkey's psychometric index at around $N \simeq 8$. If neurons are assumed to be decorrelated, the crossing occurs at $N \simeq 5$. Using the "statistical average" of the noise correlation structure, the monkey's psychometric index is approached around $N \simeq 20$.

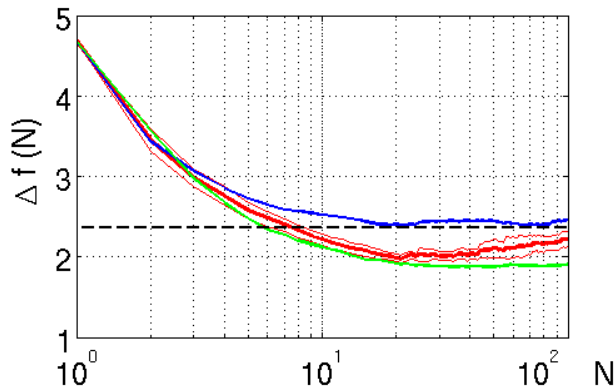

Figure 3: *Discriminability curves for various correlation structures.* Neural data: Mean firing rates over $T = 250$ msec, for $N = 100$ neurons from our recorded sample (area S1). *Green*: No noise correlations. *Red*: Random noise correlation structure (mean+std). *Blue*: Statistical average of the noise correlation structure. *Black*: Psychometric index for the monkey.

These results illustrate a number of important qualitative points. First, a known fact: the chosen noise correlation structure in a model can have a strong impact on the neural readout. Maybe not so known is the fact that considering a simplified, "statistical average" of noise correlations may lead to dramatically different results in the estimation of certain quantities such as discriminability. Thus, inferring a noise correlation structure must be done with as much care as possible in sticking to the available structure in the data. We think the method of extrapolation of noise correlation matrices proposed here offers a means to stick closer to the statistical structure (partially) observed in the data, than more simplistic methods.

Second, a comment must be made on the typical number of neurons required to attain the monkey's behavioral level of performance ($N \leq 10$ using our extrapolation method for noise correlations). No matter the exact computation and sensory modality, it is a known fact that a few sensory neurons are sufficient to convey as much information about the stimulus as the monkey seems to be using, *when their spikes are counted over long periods of time* (typically, several hundreds of ms) [13, 14]. This is paradoxical when considering the number of neurons involved, even in such a simple task as that studied here. The simplest explanation to this paradox is that this spike count over several hundreds of milliseconds is not accessible behaviorally to the animal. Most likely, the animal's percept relies on much more *instantaneous* integrations of its sensory areas' activities, so that the contributions of many more neurons are required to achieve the animal's level of accuracy. In this optic, we have started to study an alternative type of linear readout from a neural population, based on its instantaneous spiking activity, which we term 'online readout' [7]. We believe that such an approach, combined with the method proposed here to account for noise correlations with more accuracy, will lead to better approximations of the number of neurons and typical integration times used by the monkey in solving this type of task.

## 5   Conclusion

We have proposed a new method to account for the noise correlation structure in a neural population, on the basis of partial correlation data. The method is based on the statistical link between signal and noise correlation, which is a reflection of the underlying neural connectivity, and can be estimated through pairwise simultaneous recordings. Noise correlation matrices generated in accordance with this statistical link display robust properties across possible configurations, and thus provide reliable estimates for the impact of noise correlation – if, naturally, the statistical model linking signal and noise correlation is accurate enough. We applied this method to estimate the linear discriminability in $N$-tuples of neurons from area S1 when their spikes are counted over 200 msec. We found that less than 10 neurons can account for the monkey's behavioral accuracy, suggesting that percepts based on full neural populations are likely based on much shorter integration times.

# References

[1] Zohary, E. and Shadlen, M.N. and Newsome, W.T. (1994) Correlated neuronal discharge rate and its implications for psychophysical performance, *Nature* **370**(6485): 140–143

[2] Romo, R., Hernández, A., Zainos, A. and Salinas, E. (2003) Correlated neuronal discharges that increase coding efficiency during perceptual discrimination, *Neuron* **38**(4): 649–657

[3] Averbeck, B.B., Latham, P.E. and Pouget, A. (2006) Neural correlations, population coding and computation, *Nature Reviews Neuroscience* **7**(5): 358–366

[4] Abeles, M. (1991) Corticonics: Neural circuits of the cerebral cortex, *Cambridge Univ Pr*

[5] Lee, D., Port, N.L., Kruse, W. and Georgopoulos, A.P. (1998) Variability and correlated noise in the discharge of neurons in motor and parietal areas of the primate cortex, *Journal of Neuroscience* **18**(3)

[6] Petersen, R.S., Panzeri, S. and Diamond, M.E. (2001) Population coding of stimulus location in rat somatosensory cortex, *Neuron* **32**(3): 503–514

[7] Wohrer, A., Romo, R. and Machens, C. K. (2010) Online readout of frequency information in areas SI and SII *Computational and Systems Neuroscience 2010 (CoSyne)*

[8] Abbott, LF and Dayan, P. (1999) The effect of correlated variability on the accuracy of a population code, *Neural Computation* **11**(1): 91–101

[9] Horn, R.A. and Johnson, C.R. (1990) Matrix analysis, *Cambridge Univ Pr*

[10] Johnson, R.A. and Wichern, D.W. (1998) Applied multivariate statistical analysis, *Prentice Hall Englewood Cliffs, NJ*

[11] Janik, R.A. and Nowak, M.A. (2003) Wishart and anti-Wishart random matrices, *Journal of Physics A: Mathematical and General* **36**: 3629–3637

[12] Fisher, R.A. (1915) Frequency Distribution of the Values of the Correlation Coefficients in Samples from an Indefinitely Large Population, *Biometrika* **10**(4)

[13] Britten, KH, Shadlen, MN, Newsome, WT and Movshon, JA (1992) The analysis of visual motion: a comparison of neuronal and psychophysical performance, *Journal of Neuroscience* **12**(12)

[14] Romo, R. and Salinas, E. (2003) Flutter discrimination: neural codes, perception, memory and decision making, *Nature Reviews Neuroscience* **4**(3): 203–218

